# Loop Series and Bethe Variational Bounds in Attractive Graphical Models

**Erik B. Sudderth and Martin J. Wainwright**
Electrical Engineering & Computer Science, University of California, Berkeley
`sudderth@eecs.berkeley.edu, wainwrig@eecs.berkeley.edu`

**Alan S. Willsky**
Electrical Engineering & Computer Science, Massachusetts Institute of Technology
`willsky@mit.edu`

## Abstract

Variational methods are frequently used to approximate or bound the partition or likelihood function of a Markov random field. Methods based on mean field theory are guaranteed to provide lower bounds, whereas certain types of convex relaxations provide upper bounds. In general, loopy belief propagation (BP) provides often accurate approximations, but not bounds. We prove that for a class of attractive binary models, the so–called Bethe approximation associated with any fixed point of loopy BP always lower bounds the true likelihood. Empirically, this bound is much tighter than the naive mean field bound, and requires no further work than running BP. We establish these lower bounds using a loop series expansion due to Chertkov and Chernyak, which we show can be derived as a consequence of the tree reparameterization characterization of BP fixed points.

## 1 Introduction

Graphical models are widely used in many areas, including statistical machine learning, computer vision, bioinformatics, and communications. Such applications typically require computationally efficient methods for (approximately) solving various problems, including computing marginal distributions and likelihood functions. The variational framework provides a suite of candidate methods, including mean field approximations [3, 9], the sum–product or *belief propagation* (BP) algorithm [11, 14], Kikuchi and cluster variational methods [23], and related convex relaxations [21].

The likelihood or partition function of an undirected graphical model is of fundamental interest in many contexts, including parameter estimation, error bounds in hypothesis testing, and combinatorial enumeration. In rough terms, particular variational methods can be understood as solving optimization problems whose optima approximate the log partition function. For mean field methods, this optimal value is desirably guaranteed to lower bound the true likelihood [9]. For other methods, including the *Bethe variational problem* underlying loopy BP [23], optima may either over–estimate or under–estimate the truth. Although "convexified" relaxations of the Bethe problem yield upper bounds [21], to date the best known lower bounds on the partition function are based on mean field theory. Recent work has studied *loop series expansions* [2, 4] of the partition function, which generate better approximations but not, in general, bounds.

Several existing theoretical results show that loopy BP, and the corresponding Bethe approximation, have desirable properties for graphical models with long cycles [15] or sufficiently weak dependencies [6, 7, 12, 19]. However, these results do not explain the excellent empirical performance of BP in many graphs with short cycles, like the nearest–neighbor grids arising in spatial statistics and low–level vision [3, 18, 22]. Such models often encode "smoothness" priors, and thus have *attractive* interactions which encourage connected variables to share common values. The first main contribution of this paper is to demonstrate a family of attractive models for which the Bethe variational method always yields lower bounds on the true likelihood. Although we focus on models with binary variables (but arbitrary order of interactions), we suspect that some ideas are more generally applicable. For such models, these lower bounds are easily computed from any fixed point of loopy BP, and empirically improve substantially on naive mean field bounds.

Our second main contribution lies in the route used to establish the Bethe lower bounds. In particular, Sec. 3 uses the reparameterization characterization of BP fixed points [20] to provide a simple derivation for the loop series expansion of Chertkov and Chernyak [2]. The Bethe approximation is the first term in this representation of the true partition function. Sec. 4 then identifies attractive models for which all terms in this expansion are positive, thus establishing the Bethe lower bound. We conclude with empirical results demonstrating the accuracy of this bound, and discuss implications for future analysis and applications of loopy BP.

## 2 Undirected Graphical Models

Given an undirected graph $G = (V, E)$, with edges $(s, t) \in E$ connecting $n$ vertices $s \in V$, a graphical model associates each node with a random variable $X_s$ taking values $x_s \in \mathcal{X}$. For pairwise *Markov random fields* (MRFs) as in Fig. 1, the joint distribution of $x := \{x_s \mid s \in V\}$ is specified via a normalized product of local *compatibility functions*:

$$p(x) = \frac{1}{Z(\psi)} \prod_{s \in V} \psi_s(x_s) \prod_{(s,t) \in E} \psi_{st}(x_s, x_t) \tag{1}$$

The *partition function* $Z(\psi) := \sum_{x \in \mathcal{X}^n} \prod_s \psi_s(x_s) \prod_{(s,t)} \psi_{st}(x_s, x_t)$, whose value depends on the compatibilities $\psi$, is defined so that $p(x)$ is properly normalized. We also consider distributions defined by hypergraphs $G = (V, C)$, where each hyperedge $c \in C$ connects some subset of the vertices $(c \subset V)$. Letting $x_c := \{x_s \mid s \in c\}$, the corresponding joint distribution equals

$$p(x) = \frac{1}{Z(\psi)} \prod_{s \in V} \psi_s(x_s) \prod_{c \in C} \psi_c(x_c) \tag{2}$$

where as before $Z(\psi) = \sum_{x \in \mathcal{X}^n} \prod_s \psi_s(x_s) \prod_c \psi_c(x_c)$. Such higher–order random fields are conveniently described by the bipartite *factor graphs* [11] of Fig. 2.

In statistical physics, the partition function arises in the study of how physical systems respond to changes in external stimuli or temperature [23]. Alternatively, when compatibility functions are parameterized by exponential families [20], $\log Z(\psi)$ is the family's *cumulant generating function*, and thus intrinsically related to the model's marginal statistics. For directed Bayesian networks (which can be factored as in eq. (2)), $Z(\psi)$ is the marginal likelihood of observed data, and plays a central role in learning and model selection [9]. However, for general graphs coupling discrete random variables, the cost of exactly evaluating $Z(\psi)$ grows exponentially with $n$ [8]. Computationally tractable families of bounds on the true partition function are thus of great practical interest.

### 2.1 Attractive Discrete Random Fields

In this paper, we focus on binary random vectors $x \in \{0, 1\}^n$. We say that a pairwise MRF, with compatibility functions $\psi_{st} : \{0, 1\}^2 \to \mathbb{R}^+$, has *attractive* interactions if

$$\psi_{st}(0, 0) \, \psi_{st}(1, 1) \geq \psi_{st}(0, 1) \, \psi_{st}(1, 0) \tag{3}$$

for each edge $(s, t) \in E$. Intuitively, this condition requires all potentials to place greater weight on configurations where neighboring variables take the same value. Our later analysis is based on pairwise marginal distributions $\tau_{st}(x_s, x_t)$, which we parameterize as follows:

$$\tau_{st}(x_s, x_t) = \begin{bmatrix} 1 - \tau_s - \tau_t + \tau_{st} & \tau_t - \tau_{st} \\ \tau_s - \tau_{st} & \tau_{st} \end{bmatrix} \qquad \begin{aligned} \tau_s &:= \mathbb{E}_{\tau_{st}}[X_s] \\ \tau_{st} &:= \mathbb{E}_{\tau_{st}}[X_s X_t] \end{aligned} \tag{4}$$

We let $\mathbb{E}_{\tau_{st}}[\cdot]$ denote expectation with respect to $\tau_{st}(x_s, x_t)$, so that $\tau_{st}$ is the probability that $X_s = X_t = 1$. This normalized matrix is attractive, satisfying eq. (3), if and only if $\tau_{st} \geq \tau_s \tau_t$.

For binary variables, the pairwise MRF of eq. (1) provides one representation of a general, inhomogeneous *Ising model*. In the statistical physics literature, Ising models are typically expressed by coupling random spins $z_s \in \{-1, +1\}$ with symmetric potentials $\log \psi_{st}(z_s, z_t) = \theta_{st} z_s z_t$. The attractiveness condition of eq. (3) then becomes $\theta_{st} \geq 0$, and the resulting model has *ferromagnetic* interactions. Furthermore, pairwise MRFs satisfy the *regularity* condition of [10], and thus allow tractable MAP estimation via graph cuts [5], if and only if they are attractive. Even for attractive models, however, calculation of the partition function in non–planar graphs is #P–complete [8].

To define families of higher–order attractive potentials, we first consider a probability distribution $\tau_c(x_c)$ on $k = |c|$ binary variables. Generalizing eq. (4), we parameterize such distributions by the

following collection of $2^k - 1$ mean parameters:

$$\tau_a := \mathbb{E}_{\tau_c}\left[\prod_{s \in a} X_s\right] \qquad \emptyset \neq a \subseteq c \tag{5}$$

For example, $\tau_{stu}(x_s, x_t, x_u)$ would be parameterized by $\{\tau_s, \tau_t, \tau_u, \tau_{st}, \tau_{su}, \tau_{tu}, \tau_{stu}\}$. For any subset $a \subseteq c$, we then define the following central moment statistic:

$$\kappa_a := \mathbb{E}_{\tau_c}\left[\prod_{s \in a}(X_s - \tau_s)\right] \qquad \emptyset \neq a \subseteq c \tag{6}$$

Note that $\kappa_s = 0$, while $\kappa_{st} = \text{Cov}_\tau(X_s, X_t) = \tau_{st} - \tau_s \tau_t$. The third–order central moment then equals the cumulant $\kappa_{stu} = \tau_{stu} - \tau_{st}\tau_u - \tau_{su}\tau_t - \tau_{tu}\tau_s + 2\tau_s\tau_t\tau_u$.

Given these definitions, we say that a probability distribution $\tau_c(x_c)$ is *attractive* if the central moments associated with all subsets $a \subseteq c$ of binary variables are non–negative ($\kappa_a \geq 0$). Similarly, a compatibility function $\psi_c(x_c)$ is attractive if the probability distribution attained by normalizing its values has non–negative central moments. For example, the following potential is easily shown to satisfy this condition for all degrees $k = |c|$, and any scalar $\theta_c > 0$:

$$\log \psi_c(x_1, \ldots, x_k) = \begin{cases} \theta_c & x_1 = x_2 = \cdots = x_k \\ -\theta_c & \text{otherwise} \end{cases} \tag{7}$$

## 2.2 Belief Propagation and the Bethe Variational Principle

Many applications of graphical models require estimates of the posterior marginal distributions of individual variables $\tau_s(x_s)$ or factors $\tau_c(x_c)$. Loopy *belief propagation* (BP) approximates these marginals via a series of *messages* passed among nodes of the graphical model [14, 23]. Let $\Gamma(s)$ denote the set of factors which depend on $X_s$, or equivalently the neighbors of node $s$ in the corresponding factor graph. The BP algorithm then iterates the following message updates:

$$\bar{m}_{sc}(x_s) \leftarrow \psi_s(x_s) \prod_{d \in \Gamma(s) \backslash c} m_{ds}(x_s) \qquad m_{cs}(x_s) \leftarrow \sum_{x_{c \backslash s}} \psi_c(x_c) \prod_{t \in c \backslash s} \bar{m}_{tc}(x_t) \tag{8}$$

The left–hand expression updates the message $\bar{m}_{sc}(x_s)$ passed from variable node $s$ to factor $c$. New outgoing messages $m_{cs}(x_s)$ from factor $c$ to each $s \in c$ are then determined by marginalizing the incoming messages from other nodes. At any iteration, appropriately normalized products of these messages define estimates of the desired marginals:

$$\tau_s(x_s) \propto \psi_s(x_s) \prod_{c \in \Gamma(s)} m_{cs}(x_s) \qquad \tau_c(x_c) \propto \psi_c(x_c) \prod_{t \in c} \bar{m}_{tc}(x_t) \tag{9}$$

In tree–structured graphs, BP defines a dynamic programming recursion which converges to the exact marginals after finitely many iterations [11, 14]. In graphs with cycles, however, convergence is not guaranteed, and *pseudo–marginals* computed via eq. (9) are (often good) approximations.

A wide range of inference algorithms can be derived via variational approximations [9] to the true partition function. Loopy BP is implicitly associated with the following *Bethe approximation*:

$$\log Z_\beta(\psi; \tau) = \sum_{s \in V} \sum_{x_s} \tau_s(x_s) \log \psi_s(x_s) + \sum_{c \in C} \sum_{x_c} \tau_c(x_c) \log \psi_c(x_c)$$

$$- \sum_{s \in V} \sum_{x_s} \tau_s(x_s) \log \tau_s(x_s) - \sum_{c \in C} \sum_{x_c} \tau_c(x_c) \log \frac{\tau_c(x_c)}{\prod_{t \in c} \tau_t(x_t)} \tag{10}$$

Fixed points of loopy BP correspond to stationary points of this Bethe approximation [23], subject to the local marginalization constraints $\sum_{x_{c \backslash s}} \tau_c(x_c) = \tau_s(x_s)$.

## 3 Reparameterization and Loop Series Expansions

As discussed in Sec. 2.2, any BP fixed point is in one–to–one correspondence with a set $\{\tau_s, \tau_c\}$ of pseudo–marginals associated with each of the graph's nodes $s \in V$ and factors $c \in C$. These pseudo–marginals then lead to an alternative *parameterization* [20] of the factor graph of eq. (2):

$$p(x) = \frac{1}{Z(\tau)} \prod_{s \in V} \tau_s(x_s) \prod_{c \in C} \frac{\tau_c(x_c)}{\prod_{t \in c} \tau_t(x_t)} \tag{11}$$

For pairwise MRFs, the reparameterized compatibility functions equal $\tau_{st}(x_s, x_t)/\tau_s(x_s)\tau_t(x_t)$. The BP algorithm effectively searches for reparameterizations which are *tree–consistent*, so that

$\tau_c(x_c)$ is the exact marginal distribution of $X_c$ for *any* tree (or forest) embedded in the original graph [20]. In later sections, we take expectations with respect to $\tau_c(x_c)$ of functions $f(x_c)$ defined over individual factors. Although these pseudo–marginals will in general not equal the *true* marginals $p_c(x_c)$, BP fixed points ensure local consistency so that $\mathbb{E}_{\tau_c}[f(X_c)]$ is well–defined.

Using eq. (10), it is easily shown that the Bethe approximation $Z_\beta(\tau; \tau) = 1$ for *any* joint distribution defined by reparameterized potentials as in eq. (11). For simplicity, the remainder of this paper focuses on reparameterized models of this form, and analyzes properties of the corresponding exact partition function $Z(\tau)$. The resulting expansions and bounds are then related to the original MRF's partition function via the positive constant $Z(\psi)/Z(\tau) = Z_\beta(\psi; \tau)$ of eq. (10).

Recently, Chertkov and Chernyak proposed a finite *loop series expansion* [2] of the partition function, whose first term coincides with the Bethe approximation. They provide two derivations: one applies a trigonometric identity to Fourier representations of binary variables, while the second employs a saddle point approximation obtained via an auxiliary field of complex variables. The *gauge transformations* underlying these derivations are a type of reparameterization, but their form is complicated by auxiliary variables and extraneous degrees of freedom. In this section, we show that the fixed point characterization of eq. (11) leads to a more direct, and arguably simpler, derivation.

### 3.1 Pairwise Loop Series Expansions

We begin by developing a loop series expansion for pairwise MRFs. Given an undirected graph $G = (V, E)$, and some subset $F \subseteq E$ of the graph's edges, let $d_s(F)$ denote the degree (number of neighbors) of node $s$ in the subgraph induced by $F$. As illustrated in Fig. 1, any subset $F$ for which all nodes $s \in V$ have degree $d_s(F) \neq 1$ defines a *generalized loop* [2]. The partition function for any binary, pairwise MRF can then be expanded via an associated set of *loop corrections*.

**Proposition 1.** *Consider a pairwise MRF defined on an undirected $G = (V, E)$, with reparameterized potentials as in eq.* (11). *The associated partition function then equals*

$$Z(\tau) = 1 + \sum_{\emptyset \neq F \subseteq E} \boldsymbol{\beta}_F \prod_{s \in V} \mathbb{E}_{\tau_s}\left[(X_s - \tau_s)^{d_s(F)}\right] \qquad \boldsymbol{\beta}_F := \prod_{(s,t) \in F} \beta_{st} \qquad (12)$$

$$\beta_{st} := \frac{\tau_{st} - \tau_s \tau_t}{\tau_s(1 - \tau_s)\tau_t(1 - \tau_t)} = \frac{\mathrm{Cov}_{\tau_{st}}(X_s, X_t)}{\mathrm{Var}_{\tau_s}(X_s)\,\mathrm{Var}_{\tau_t}(X_t)} \qquad (13)$$

*where only generalized loops $F$ lead to non–zero terms in the sum of eq.* (12), *and*

$$\mathbb{E}_{\tau_s}\left[(X_s - \tau_s)^d\right] = \tau_s(1 - \tau_s)\left[(1 - \tau_s)^{d-1} + (-1)^d (\tau_s)^{d-1}\right] \qquad (14)$$

*are central moments of the binary variables at individual nodes.*

*Proof.* To establish the expansion of eq. (12), we exploit the following polynomial representation of reparameterized pairwise compatibility functions:

$$\frac{\tau_{st}(x_s, x_t)}{\tau_s(x_s)\tau_t(x_t)} = 1 + \beta_{st}(x_s - \tau_s)(x_t - \tau_t) \qquad (15)$$

As verified in [17], this expression is satisfied for any $(x_s, x_t) \in \{0, 1\}^2$ if $\beta_{st}$ is defined as in eq. (13). For attractive models satisfying eq. (3), $\beta_{st} \geq 0$ for all edges. Using $\mathbb{E}_{\tilde{\tau}}[\cdot]$ to denote expectation with respect to the fully factorized distribution $\tilde{\tau}(x) = \prod_s \tau_s(x_s)$, we then have

$$Z(\tau) = \sum_{x \in \{0,1\}^n} \prod_{s \in V} \tau_s(x_s) \prod_{(s,t) \in E} \frac{\tau_{st}(x_s, x_t)}{\tau_s(x_s)\tau_t(x_t)}$$

$$= \mathbb{E}_{\tilde{\tau}}\left[\prod_{(s,t) \in E} \frac{\tau_{st}(X_s, X_t)}{\tau_s(X_s)\tau_t(X_t)}\right] = \mathbb{E}_{\tilde{\tau}}\left[\prod_{(s,t) \in E} 1 + \beta_{st}(X_s - \tau_s)(X_t - \tau_t)\right] \qquad (16)$$

Expanding this polynomial via the expectation operator's linearity, we recover one term for each non–empty subset $F \subseteq E$ of the graph's edges:

$$Z(\tau) = 1 + \sum_{\emptyset \neq F \subseteq E} \mathbb{E}_{\tilde{\tau}}\left[\prod_{(s,t) \in F} \beta_{st}(X_s - \tau_s)(X_t - \tau_t)\right] \qquad (17)$$

The expression in eq. (12) then follows from the independence structure of $\tilde{\tau}(x)$, and standard formulas for the moments of Bernoulli random variables. To evaluate these terms, note that if $d_s(F) = 1$, it follows that $\mathbb{E}_{\tau_s}[X_s - \tau_s] = 0$. There is thus one loop correction for each generalized loop $F$, in which all connected nodes have degree at least two. $\square$

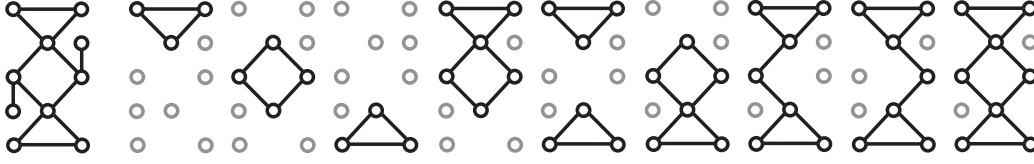

Figure 1: A pairwise MRF coupling ten binary variables (left), and the nine generalized loops in its loop series expansion (right). For attractive potentials, two of the generalized loops may have negative signs (second & third from right), while the core graph of Thm. 1 contains eight variables (far right).

Figure 1 illustrates the set of generalized loops associated with a particular pairwise MRF. These loops effectively define corrections to the Bethe estimate $Z(\tau) \approx 1$ of the partition function for reparameterized models. Tree–structured graphs do not contain any non–trivial generalized loops, and the Bethe variational approximation is thus exact.

The loop expansion formulas of [2] can be precisely recovered by transforming binary variables to a spin representation, and refactoring terms from the denominator of edge weights $\beta_{st}$ to adjacent vertices. Explicit computation of these loop corrections is in general intractable; for example, fully connected graphs with $n \geq 5$ nodes have more than $2^n$ generalized loops. In some cases, accounting for a small set of significant loop corrections may lead to improved approximations to $Z(\psi)$ [4], or more accurate belief estimates for LDPC codes [1]. We instead use the series expansion of Prop. 1 to establish analytic properties of BP fixed points.

### 3.2 Factor Graph Loop Series Expansions

We now extend the loop series expansion to higher–order MRFs defined on hypergraphs $G = (V, C)$. Let $E = \{(s, c) \mid c \in C, s \in c\}$ denote the set of edges in the factor graph representation of this MRF. As illustrated in Fig. 2, we define a generalized loop to be a subset $F \subseteq E$ of edges such that all connected factor and variable nodes have degree at least two.

**Proposition 2.** *Consider any factor graph $G = (V, C)$ with reparameterized potentials as in eq. (11), and associated edges $E$. The partition function then equals*

$$Z(\tau) = 1 + \sum_{\emptyset \neq F \subseteq E} \boldsymbol{\beta}_F \prod_{s \in V} \mathbb{E}_{\tau_s}\left[ (X_s - \tau_s)^{d_s(F)} \right] \qquad \boldsymbol{\beta}_F := \prod_{c \in C} \beta_{a_c(F)} \qquad (18)$$

$$\beta_a := \frac{\kappa_a}{\prod_{t \in a} \tau_t (1 - \tau_t)} = \frac{\mathbb{E}_{\tau_c}\left[ \prod_{s \in a} (X_s - \tau_s) \right]}{\prod_{t \in a} \mathrm{Var}_{\tau_t}(X_t)} \qquad (19)$$

*where $a_c(F) := \{s \in c \mid (s, c) \in F\}$ denotes the subset of variables linked to factor node $c$ by the edges in $F$. Only generalized loops $F$ lead to non–zero terms in the sum of eq. (18).*

*Proof.* As before, we employ a polynomial representation of the reparameterized factors in eq. (11):

$$\frac{\tau_c(x_c)}{\prod_{t \in c} \tau_t(x_t)} = 1 + \sum_{a \subseteq c, |a| \geq 2} \beta_a \prod_{s \in a} (x_s - \tau_s) \qquad (20)$$

For factor graphs with attractive reparameterized potentials, the constant $\beta_a \geq 0$ for all $a \subseteq c$. Note that this representation, which is derived in [17], reduces to that of eq. (15) when $c = \{s, t\}$. Single–variable subsets are excluded in eq. (20) because $\kappa_s = \mathbb{E}_{\tau_s}[X_s - \tau_s] = 0$.

Applying eq. (20) as in our earlier derivation for pairwise MRFs (see eq. (16)), we may express the partition function of the reparameterized factor graph as follows:

$$Z(\tau) = \mathbb{E}_{\tilde{\tau}}\left[ \prod_{c \in C} \frac{\tau_c(X_c)}{\prod_{t \in c} \tau_t(X_t)} \right] = \mathbb{E}_{\tilde{\tau}}\left[ \prod_{c \in C} 1 + \sum_{\emptyset \neq a \subseteq c} \beta_a \prod_{s \in a} (X_s - \tau_s) \right] \qquad (21)$$

Note that $\beta_a = 0$ for any subset where $|a| = 1$. There is then a one–to–one correspondence between variable node subsets $a \subseteq c$, and subsets $\{(s, c) \mid s \in a\}$ of the factor graph's edges $E$. Expanding this expression by $F \subseteq E$, it follows that each factor $c \in C$ contributes a term corresponding to the chosen subset $a_c(F)$ of its edges:

$$Z(\tau) = 1 + \sum_{\emptyset \neq F \subseteq E} \mathbb{E}_{\tilde{\tau}}\left[ \prod_{c \in C} \beta_{a_c(F)} \prod_{s \in a_c(F)} (X_s - \tau_s) \right] \qquad (22)$$

Note that $\beta_\emptyset = 1$. Equation (18) then follows from the independence properties of $\tilde{\tau}(x)$. For a term in this loop series to be non–zero, there must be no degree one variables, since $\mathbb{E}_{\tau_s}[X_s - \tau_s] = 0$. In addition, the definition of $\beta_a$ implies that there can be no degree one factor nodes. $\square$

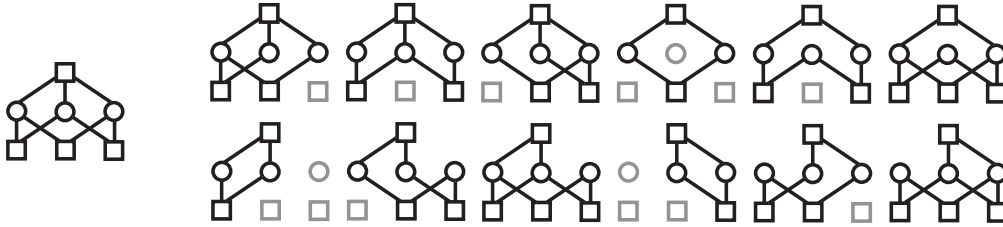

Figure 2: A factor graph (left) with three binary variables (circles) and four factor nodes (squares), and the thirteen generalized loops in its loop series expansion (right, along with the full graph).

## 4  Lower Bounds in Attractive Binary Models

The Bethe approximation underlying loopy BP differs from mean field methods [9], which *lower bound* the true log partition function $Z(\psi)$, in two key ways. First, while the Bethe entropy (second line of eq. (10)) is exact for tree–structured graphs, it *approximates* (rather than bounds) the true entropy in graphs with cycles. Second, the marginalization condition imposed by loopy BP *relaxes* (rather than strengthens) the global constraints characterizing valid distributions [21]. Nevertheless, we now show that for a large family of attractive graphical models, the Bethe approximation $Z_\beta(\psi; \tau)$ of eq. (10) lower bounds $Z(\psi)$. In contrast with mean field methods, these bounds hold only at appropriate BP fixed points, *not* for arbitrarily chosen pseudo–marginals $\tau_c(x_c)$.

### 4.1  Partition Function Bounds for Pairwise Graphical Models

Consider a pairwise MRF defined on $G = (V, E)$, as in eq. (1). Let $V_H \subseteq V$ denote the set of nodes which either belong to some cycle in $G$, or lie on a path (sequence of edges) connecting two cycles. We then define the *core graph* $H = (V_H, E_H)$ as the node–induced subgraph obtained by discarding edges from nodes outside $V_H$, so that $E_H = \{(s,t) \in E \mid s, t \in V_H\}$. The unique core graph $H$ underlying any graph $G$ can be efficiently constructed by iteratively pruning degree one nodes, or leaves, until all remaining nodes have two or more neighbors. The following theorem identifies conditions under which all terms in the loop series expansion must be non–negative.

**Theorem 1.** *Let $H = (V_H, E_H)$ be the core graph for a pairwise binary MRF, with attractive potentials satisfying eq. (3). Consider any BP fixed point for which all nodes $s \in V_H$ with three or more neighbors in $H$ have marginals $\tau_s \leq \frac{1}{2}$ (or equivalently, $\tau_s \geq \frac{1}{2}$). The corresponding Bethe variational approximation $Z_\beta(\psi; \tau)$ then lower bounds the true partition function $Z(\psi)$.*

*Proof.* It is sufficient to show that $Z(\tau) \geq 1$ for any reparameterized pairwise MRF, as in eq. (11). From eq. (9), note that loopy BP estimates the pseudo–marginal $\tau_{st}(x_s, x_t)$ via the product of $\psi_{st}(x_s, x_t)$ with message functions of *single* variables. For this reason, attractive pairwise compatibilities always lead to BP fixed points with attractive pseudo–marginals satisfying $\tau_{st} \geq \tau_s \tau_t$.

Consider the pairwise loop series expansion of eq. (12). As shown by eq. (13), attractive models lead to edge weights $\beta_{st} \geq 0$. It is thus sufficient to show that $\prod_s \mathbb{E}_{\tau_s}\big[(X_s - \tau_s)^{d_s(F)}\big] \geq 0$ for each generalized loop $F \subseteq E$. Suppose first that the graph has a single cycle, and thus exactly one non–zero generalized loop $F$. Because all connected nodes in this cycle have degree two, the bound follows because $\mathbb{E}_{\tau_s}\big[(X_s - \tau_s)^2\big] \geq 0$. More generally, we clearly have $Z(\tau) \geq 1$ in graphs where every generalized loop $F$ associates an even number of neighbors $d_s(F)$ with each node.

Focusing on generalized loops containing nodes with odd degree $d \geq 3$, eq. (14) implies that $\mathbb{E}_{\tau_s}\big[(X_s - \tau_s)^d\big] \geq 0$ for marginals satisfying $1 - \tau_s \geq \tau_s$. For BP fixed points in which $\tau_s \leq \frac{1}{2}$ for all nodes, we thus have $Z(\tau) \geq 1$. In particular, the symmetric fixed point $\tau_s = \frac{1}{2}$ leads to uniformly positive generalized loop corrections. More generally, the marginals of nodes $s$ for which $d_s(F) \leq 2$ for every generalized loop $F$ do not influence the expansion's positivity. Theorem 1 discards these nodes by examining the topology of the core graph $H$ (see Fig. 1 for an example). For fixed points where $\tau_s \geq \frac{1}{2}$ for all nodes, we rewrite the polynomial in the loop expansion of eq. (15) as $(1 + \beta_{st}(\tau_s - x_s)(\tau_t - x_t))$, and employ an analogous line of reasoning. $\qquad\square$

In addition to establishing Thm. 1, our arguments show that the true partition function *monotonically* increases as additional edges, with attractive reparameterized potentials as in eq. (11), are added to a graph with fixed pseudo–marginals $\tau_s \leq \frac{1}{2}$. For such models, the accumulation of particular loop corrections, as explored by [4], produces a sequence of increasingly tight bounds on $Z(\psi)$. In addition, we note that the conditions required by Thm. 1 are similar to those underlying classical

*correlation inequalities* [16] from the statistical physics literature. Indeed, the Griffiths–Kelly–Sherman (GKS) inequality leads to an alternative proof in cases where $\tau_s = \frac{1}{2}$ for all nodes.

For attractive Ising models in which some nodes have marginals $\tau_s > \frac{1}{2}$ and others $\tau_t < \frac{1}{2}$, the loop series expansion may contain negative terms. For small graphs like that in Fig. 1, it is possible to use *upper* bounds on the edge weights $\beta_{st}$, which follow from $\tau_{st} \leq \min(\tau_s, \tau_t)$, to cancel negative loop corrections with larger positive terms. As confirmed by the empirical results in Sec. 4.3, the lower bound $Z(\psi) \geq Z_\beta(\psi; \tau)$ thus continues to hold for many (perhaps all) attractive Ising models with less homogeneous marginal biases.

### 4.2 Partition Function Bounds for Factor Graphs

Given a factor graph $G = (V, C)$ relating binary variables, define a core graph $H = (V_H, C_H)$ by excluding variable and factor nodes which are not members of any generalized loops. As in Sec. 2.2, let $\Gamma(s)$ denote the set of factor nodes neighboring variable node $s$ in the core graph $H$.

**Theorem 2.** *Let $H = (V_H, C_H)$ be the core graph for a binary factor graph, and consider an attractive BP fixed point for which one of the following conditions holds:*

*(i) $\tau_s \leq \frac{1}{2}$ for all nodes $s \in V_H$ with $|\Gamma(s)| \geq 3$, and $\kappa_a \geq 0$ for all $a \subseteq c$, $c \in C_H$.*

*(ii) $\tau_s \geq \frac{1}{2}$ for all nodes $s \in V_H$ with $|\Gamma(s)| \geq 3$, and $(-1)^{|a|}\kappa_a \geq 0$ for all $a \subseteq c$, $c \in C_H$.*

*The Bethe approximation $Z_\beta(\psi; \tau)$ then lower bounds the true partition function $Z(\psi)$.*

For the case where $\tau_s \leq \frac{1}{2}$, the proof of this theorem is a straightforward generalization of the arguments in Sec. 4.1. When $\tau_s \geq \frac{1}{2}$, we replace all $(x_s - \tau_s)$ terms by $(\tau_s - x_s)$ in the expansion of eq. (20), and again recover uniformly positive loop corrections.

For any given BP fixed point, the conditions of Thm. 2 are easy to verify. For factor graphs, it is more challenging to determine which compatibility functions $\psi_c(x_c)$ necessarily lead to attractive fixed points. For symmetric potentials as in eq. (7), however, one can show that the conditions on $\kappa_a, a \subseteq c$ are necessarily satisfied whenever all variable nodes $s \in V_H$ have the same bias.

### 4.3 Empirical Comparison of Mean Field and Bethe Lower Bounds

In this section, we compare the accuracy of the Bethe variational bounds established by Thm. 1 to those produced by a naive, fully factored mean field approximation [3, 9]. Using the spin representation $z_s \in \{-1, +1\}$, we examine Ising models with attractive pairwise potentials $\log \psi_{st}(z_s, z_t) = \theta_{st} z_s z_t$ of varying strengths $\theta_{st} \geq 0$. We first examine a 2D torus, with potentials of uniform strength $\theta_{st} = \bar{\theta}$ and no local observations. For such MRFs, the exact partition function may be computed via Onsager's classical eigenvector method [13]. As shown in Fig. 3(a), for moderate $\bar{\theta}$ the Bethe bound $Z_\beta(\psi; \tau)$ is substantially tighter than mean field. For large $\bar{\theta}$, only two states (all spins "up" or "down") have significant probability, so that $Z(\psi) \approx 2\exp(\bar{\theta}|E|)$. In this regime, loopy BP exhibits "symmetry breaking" [6], and converges to one of these states at random with corresponding bound $Z_\beta(\psi; \tau) \approx \exp(\bar{\theta}|E|)$. As verified in Fig. 3(a), as $\bar{\theta} \to \infty$ the difference $\log Z(\psi) - \log Z_\beta(\psi; \tau) \approx \log 2 \approx 0.69$ thus remains bounded.

We also consider a set of random $10 \times 10$ nearest–neighbor grids, with inhomogeneous pairwise potentials sampled according to $|\theta_{st}| \sim \mathcal{N}(0, \bar{\theta}^2)$, and observation potentials $\log \psi_s(z_s) = \theta_s z_s$, $|\theta_s| \sim \mathcal{N}(0, 0.1^2)$. For each candidate $\bar{\theta}$, we sample 100 random MRFs, and plot the average difference $\log Z_\beta(\psi; \tau) - \log Z(\psi)$ between the true partition function and the BP (or mean field) fixed point reached from a random initialization. Fig. 3(b) first considers MRFs where $\theta_s > 0$ for all nodes, so that the conditions of Thm. 1 are satisfied for all BP fixed points. For these models, the Bethe bound is *extremely* accurate. In Fig. 3(c), we also consider MRFs where the observation potentials $\theta_s$ are of mixed signs. Although this sometimes leads to BP fixed points with negative associated loop corrections, the Bethe variational approximation nevertheless *always* lower bounds the true partition function in these examples. We hypothesize that this bound in fact holds for all attractive, binary pairwise MRFs, regardless of the observation potentials.

## 5 Discussion

We have provided an alternative, direct derivation of the partition function's loop series expansion, based on the reparameterization characterization of BP fixed points. We use this expansion to prove that the Bethe approximation lower bounds the true partition function in a family of binary attractive

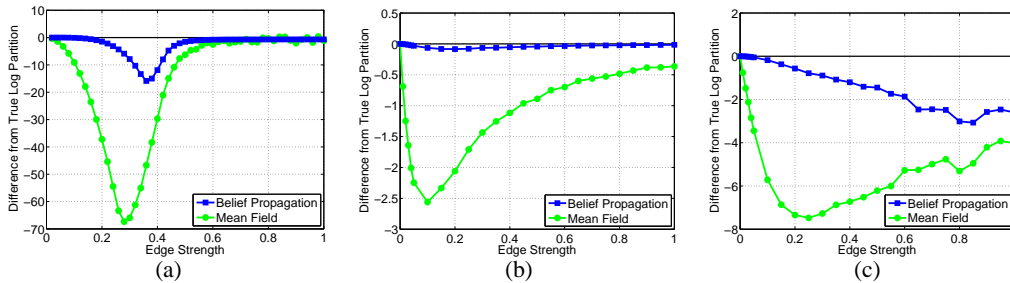

Figure 3: Bethe (dark blue, top) and naive mean field (light green, bottom) lower bounds on $\log Z(\psi)$ for three families of attractive, pairwise Ising models. (a) $30 \times 30$ torus with no local observations and homogeneous potentials. (b) $10 \times 10$ grid with random, inhomogeneous potentials and all pseudo–marginals $\tau_s > \frac{1}{2}$, satisfying the conditions of Thm. 1. (c) $10 \times 10$ grid with random, inhomogeneous potentials and pseudo–marginals of mixed biases. Empirically, the Bethe lower bound also holds for these models.

models. These results have potential implications for the suitability of loopy BP in approximate parameter estimation [3], as well as its convergence dynamics. We are currently exploring generalizations of our results to other families of attractive, or "nearly" attractive, graphical models.

**Acknowledgments** The authors thank Yair Weiss for suggesting connections to loop series expansions, and helpful conversations. Funding provided by Army Research Office Grant W911NF-05-1-0207, National Science Foundation Grant DMS-0528488, and NSF Career Grant CCF-0545862.

## References

[1] M. Chertkov and V. Y. Chernyak. Loop calculus helps to improve belief propagation and linear programming decodings of low density parity check codes. In *Allerton Conf.*, 2006.

[2] M. Chertkov and V. Y. Chernyak. Loop series for discrete statistical models on graphs. *J. Stat. Mech.*, 2006:P06009, June 2006.

[3] B. J. Frey and N. Jojic. A comparison of algorithms for inference and learning in probabilistic graphical models. *IEEE Trans. PAMI*, 27(9):1392–1416, Sept. 2005.

[4] V. Gómez, J. M. Mooij, and H. J. Kappen. Truncating the loop series expansion for BP. *JMLR*, 8:1987–2016, 2007.

[5] D. M. Greig, B. T. Porteous, and A. H. Seheult. Exact maximum a posteriori estimation for binary images. *J. R. Stat. Soc. B*, 51(2):271–279, 1989.

[6] T. Heskes. On the uniqueness of loopy belief propagation fixed points. *Neural Comp.*, 16:2379–2413, 2004.

[7] A. T. Ihler, J. W. Fisher, and A. S. Willsky. Loopy belief propagation: Convergence and effects of message errors. *JMLR*, 6:905–936, 2005.

[8] M. Jerrum and A. Sinclair. Polynomial-time approximation algorithms for the Ising model. *SIAM J. Comput.*, 22(5):1087–1116, Oct. 1993.

[9] M. I. Jordan, Z. Ghahramani, T. S. Jaakkola, and L. K. Saul. An introduction to variational methods for graphical models. *Machine Learning*, 37:183–233, 1999.

[10] V. Kolmogorov and R. Zabih. What energy functions can be minimized via graph cuts? *IEEE Trans. PAMI*, 26(2):147–159, Feb. 2004.

[11] F. R. Kschischang, B. J. Frey, and H.-A. Loeliger. Factor graphs and the sum–product algorithm. *IEEE Trans. IT*, 47(2):498–519, Feb. 2001.

[12] J. M. Mooij and H. J. Kappen. Sufficient conditions for convergence of loopy belief propagation. In *UAI 21*, pages 396–403. AUAI Press, 2005.

[13] L. Onsager. Crystal statistics I: A two–dimensional model with an order–disorder transition. *Physical Review*, 65:117–149, 1944.

[14] J. Pearl. *Probabilistic Reasoning in Intelligent Systems*. Morgan Kaufman, San Mateo, 1988.

[15] T. J. Richardson and R. L. Urbanke. The capacity of low-density parity-check codes under message-passing decoding. *IEEE Trans. IT*, 47(2):599–618, Feb. 2001.

[16] S. B. Shlosman. Correlation inequalities and their applications. *J. Math. Sci.*, 15(2):79–101, Jan. 1981.

[17] E. B. Sudderth, M. J. Wainwright, and A. S. Willsky. Loop series and Bethe variational bounds in attractive graphical models. UC Berkeley, EECS department technical report, in preparation, 2008.

[18] M. F. Tappen and W. T. Freeman. Comparison of graph cuts with belief propagation for stereo, using identical MRF parameters. In *ICCV*, volume 2, pages 900–907, 2003.

[19] S. C. Tatikonda and M. I. Jordan. Loopy belief propagation and Gibbs measures. In *UAI 18*, pages 493–500. Morgan Kaufmann, 2002.

[20] M. J. Wainwright, T. S. Jaakkola, and A. S. Willsky. Tree–based reparameterization framework for analysis of sum–product and related algorithms. *IEEE Trans. IT*, 49(5):1120–1146, May 2003.

[21] M. J. Wainwright, T. S. Jaakkola, and A. S. Willsky. A new class of upper bounds on the log partition function. *IEEE Trans. IT*, 51(7):2313–2335, July 2005.

[22] Y. Weiss. Comparing the mean field method and belief propagation for approximate inference in MRFs. In D. Saad and M. Opper, editors, *Advanced Mean Field Methods*. MIT Press, 2001.

[23] J. S. Yedidia, W. T. Freeman, and Y. Weiss. Constructing free energy approximations and generalized belief propagation algorithms. *IEEE Trans. IT*, 51(7):2282–2312, July 2005.

